# Predicting spike times from subthreshold dynamics of a neuron

**Ryota Kobayashi**
Department of Physics
Kyoto University
Kyoto 606-8502, Japan
`kobayashi@ton.scphys.kyoto-u.ac.jp`

**Shigeru Shinomoto**
Department of Physics
Kyoto University
Kyoto 606-8502, Japan
`shinomoto@scphys.kyoto-u.ac.jp`

## Abstract

It has been established that a neuron reproduces highly precise spike response to identical fluctuating input currents. We wish to accurately predict the firing times of a given neuron for any input current. For this purpose we adopt a model that mimics the dynamics of the membrane potential, and then take a cue from its dynamics for predicting the spike occurrence for a novel input current. It is found that the prediction is significantly improved by observing the state space of the membrane potential and its time derivative(s) in advance of a possible spike, in comparison to simply thresholding an instantaneous value of the estimated potential.

## 1 Introduction

Since Hodgkin and Huxley [1] described the ionic flux across the neuronal membrane with four nonlinear differential equations more than half a century ago, continuous efforts have been made either to extract an essence of the nonlinear dynamical aspect by simplifying the model, or construct ever more realistic models by including more ionic channels in the model.

In the simplification proposed by FitzHugh [2] and Nagumo et al [3], the number of equations is reduced to two: the fast and slow variables which minimally represent the excitable dynamics. The leaky integrate-and-fire model [4], originally proposed far in advance of the Hodgkin-Huxley model, consists of only one variable that corresponds to the membrane potential, with a voltage resetting mechanism. Those simplified models have been successful in not only extracting the essence of the dynamics, but also in reducing the computational cost of studying the large-scale dynamics of an assembly of neurons.

In contrast to such taste for simplification, there are also a number of studies that pursue realism by developing multi-compartment models and installing newly found ionic channels. User-friendly simulation platforms, such as NEURON [5] and GENESIS [6], enable experimental neurophysiologists to reproduce casually their experimental results or to explore potentially interesting phenomena for a new experiment to be performed.

Though those simulators have been successful in reproducing qualitative aspects of neuronal responses to various conditions, quantitative reproduction as well as prediction for novel experiments

appears to be difficult to realize [7]. The difficulty is due to the complexity of the model accompanied with a large number of undetermined free parameters. Even if a true model of a particular neuron is included in the family of models, it is practically difficult to explore the true parameters in the high-dimensional space of parameters that dominate the nonlinear dynamics.

Recently it was suggested by Kistler et al [8, 9] to extend the leaky integrate-and-fire model so that real membrane dynamics of any neuron can be adopted. The so called "spike response model" has been successful in not only reproducing the data but also in predicting the spike timing for a novel input current [8, 9, 10, 11]. The details of an integration kernel are learned easily from the sample data. The fairly precise prediction achieved by such a simple model indicates that the spike occurrence is determined principally by the subthreshold dynamics. In other words, the highly nonlinear dynamics of a neuron can be decomposed into two simple, predictable processes: a relatively simple subthreshold dynamics, and the dynamics of an action potential of a nearly fixed shape (Fig.1).

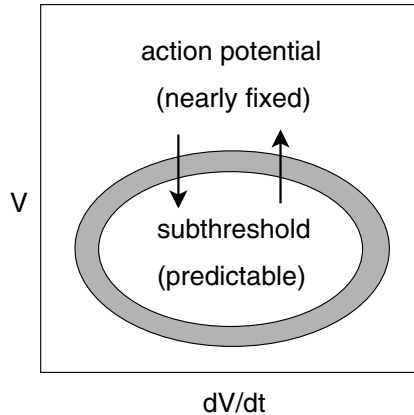

Figure 1: The highly nonlinear dynamics of a neuron is decomposed into two simple, predictable processes.

In this paper, we propose a framework of improving the prediction of spike times by paying close attention to the transfer between the two predictable processes mentioned above. It is assumed in the original spike response model that a spike occurs if the membrane potential exceeds a certain threshold [9]. We revised this rule to maximally utilize the information of a higher-dimensional state space, consisting of not only the instantaneous membrane potential, but also its time derivative(s). Such a subthreshold state can provide cues for the occurrence of a spike, but with a certain time difference. For the purpose of exploring the optimal time shift, we propose a method of maximizing the mutual information between the subthreshold state and the occurrence of a spike.

By employing the linear filter model [12] and the spike response model [9] for mimicking the subthreshold voltage response of a neuron, we examine how much the present framework may improve the prediction for simulation data of the fast-spiking model [13].

## 2 Methods

The response of a neuron is precisely reproduced when presented with identical fluctuating input currents [14]. This implies that the neuronal membrane potential $V(t)$ is determined by the past input current $\{I(t)\}$, or

$$V(t) = F(\{I(t)\}), \qquad (1)$$

where $F(\{I(t)\})$ represents a functional of a time-dependent current $I(t)$. A rapid swing in the polarity of the membrane potential is called a "spike." The occurrence of a spike could be defined practically by measuring the membrane potential $V(t)$ exceeding a certain threshold,

$$V(t) > V_{\text{th}}. \qquad (2)$$

The time of each spike could be defined either as the first time the threshold is exceeded, or as the peak of the action potential that follows the crossing.

Kistler et al [8] and Jolivet et al [10, 11] proposed a method of mimicking the membrane dynamics of a target neuron with the simple spike response model in which an input current is linearly integrated. The leaky integrate-and-fire model can be regarded as an example of the spike response model [9]; the differential equation can be rewritten as an integral equation in which the membrane potential is given as the integral of the past input current with an exponentially decaying kernel. The spike response model is an extension of the leaky integrate-and-fire model, where the integrating kernel is adaptively determined by the data, and the after hyperpolarizing potential is added subsequently to every spike. It is also possible to further include terms that reduce the responsiveness and increase the threshold after an action potential takes place.

Even in the learning stage, no model is able to perfectly reproduce the output $V(t)$ of a target neuron for a given input $I(t)$. We will denote the output of the model (in the lower case) as

$$v(t) = f_k(\{I(t)\}), \tag{3}$$

where $k$ represents a set of model parameters. The model parameters are learned by mimicking sample input-output data. This is achieved by minimizing the integrated square error,

$$E_k = \int (V(t) - v(t))^2 \, dt. \tag{4}$$

## 2.1 State space method

As the output of the model $v(t)$ is not identical to the true membrane potential of the target neuron $V(t)$, a spike occurrence cannot be determined accurately by simply applying the same threshold rule Eq.(2) to $v(t)$. In this paper, we suggest revising the spike generation rule so that a spike occurrence is best predicted from the model potential $v(t)$.

Suppose that we have adjusted the parameters of $f_k(\{I(t)\})$ so that the output of the model $\{v(t)\}$ best approximates the membrane potential $\{V(t)\}$ of a target neuron for a given set of currents $\{I(t)\}$. If the sample data set $\{I(t), V(t)\}$ employed in learning is large enough, the spike occurrence can be predicted by estimating an empirical probability of a spike being generated at the time $t$, given a time-dependent orbit of an estimated output, $\{v(t)\}$, as

$$p_{\text{spike}}(t|\{v(t)\}). \tag{5}$$

In a practical experiment, however, the amount of collectable data is insufficient for estimating the spiking probability with respect to any orbit of $v(t)$. In place of such exhaustive examination, we suggest utilizing the state space information such as the time derivatives of the model potential at a certain time. The spike occurrence at time $t$ could be predicted from the $m$-dimensional state space information $\vec{v} \equiv (v, v', \cdots, v^{(m-1)})$, as observed at a time $s$ before $t$, as

$$p_{\text{spike}}(t|\vec{v}_{t-s}), \tag{6}$$

where $\vec{v}_{t-s} \equiv (v(t-s), v'(t-s), \cdots, v^{(m-1)}(t-s))$.

## 2.2 Determination of the optimal time shift

The time shift $s$ introduced in the spike time prediction, Eq.(6), is chosen to make the prediction more reliable. We propose optimizing the time shift $s$ by maximizing the mutual information between the state space information $\vec{v}_{t-s}$ and the presence or absence of a spike at a time interval $(t - \delta t/2, t + \delta t/2]$, which is denoted as $z_t = 1$ or $0$. The mutual information [15] is given as

$$MI(z_t; \vec{v}_{t-s}) = MI(\vec{v}_{t-s}; z_t) = H(\vec{v}_{t-s}) - H(\vec{v}_{t-s}|z_t), \tag{7}$$

where

$$H(\vec{v}_{t-s}) = -\int d\vec{v}_{t-s}\, p(\vec{v}_{t-s}) \log p(\vec{v}_{t-s}), \tag{8}$$

$$H(\vec{v}_{t-s}|z_t) = -\sum_{z_t \in \{0,1\}} \int d\vec{v}_{t-s}\, p(\vec{v}_{t-s}|z_t) p(z_t) \log p(\vec{v}_{t-s}|z_t). \tag{9}$$

Here, $p(\vec{v}_{t-s}|z_t)$ is the probability, given the presence or absence of a spike at time $t \in (t - \delta t/2, t + \delta t/2]$, of the state being $\vec{v}_{t-s}$, a time $s$ before the spike.

With the time difference $s$ optimized, we then obtain the empirical probability of the spike occurrence at the time $t$, given the state space information at the time $t - s$, using the Bayes theorem,

$$p_{\text{spike}}(t|\vec{v}_{t-s}) \propto p(z_t = 1|\vec{v}_{t-s}) = \frac{p(\vec{v}_{t-s}|z_t)p(z_t)}{p(\vec{v}_{t-s})}. \qquad (10)$$

## 3   Results

We evaluated our state space method of predicting spike times by applying it to target data obtained for a fast-spiking neuron model proposed by Erisir et al [13] (see Appendix). In this virtual experiment, two fluctuating currents characterized by the same mean and standard deviation are injected to the (model) fast-spiking neuron to obtain two sets of input-output data $\{I(t), V(t)\}$. A predictive model was trained using one sample data set, and then its predictive performance for the other sample data was evaluated.

Each input current is generated by the Ornstein-Uhlenbeck process. We have tested two kinds of fluctuating currents characterized with different means and standard deviations: (Currents I) the mean $\mu = 1.5$ [$\mu$A], the standard deviation $\sigma = 1.0$ [$\mu$A] and the time scale of the fluctuation $\tau = 2$ [msec]; (Currents II) the mean $\mu = 0.0$ [$\mu$A], the standard deviation $\sigma = 4.0$ [$\mu$A] and the time scale of the fluctuation $\tau = 2$ [msec]. For each set with these statistics, we derived two independent sequences of $I(t)$. In this study we adopted the linear filter model and the spike response model as prediction models.

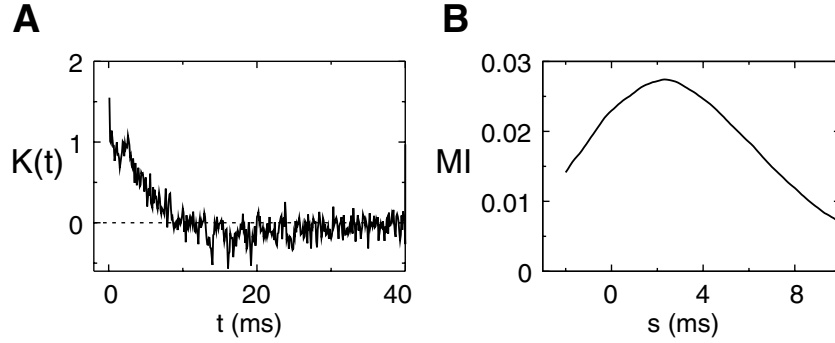

Figure 2: A: The estimated kernel. B: The mutual information between the estimated potential and the occurrence of a spike.

We briefly describe here the results for the linear filter model [12],

$$v(t) = \int_0^\infty K(t')I(t - t')\, dt' + v_0. \qquad (11)$$

The model parameters $k$ consist of the shape of the kernel $K(t)$ and the constant $v_0$. In learning the target sample data $\{I(t), V(t)\}$, these parameters are adjusted to minimize the integrated square error, Eq.(4). Figure 2A depicts the shape of the kernel $K(t)$ estimated from the target sample data $\{I(t), V(t)\}$ obtained from the virtual experiment of the fast-spiking neuron model.

Based on the voltage estimation $v(t)$ with respect to sample data, we compute the empirical probabilities, $p(\vec{v}_{t-s})$, $p(\vec{v}_{t-s}|z_t)$ and $p(z_t)$ for two-dimensional state space information $\vec{v}_{t-s} \equiv (v(t-s), v'(t-s))$. In computing empirical probabilities, we quantized the two-dimensional phase space $\vec{v} \equiv (v, v')$, and the time. In the discretized time, we counted the occurrence of a spike, $z_t = 1$, for the bins in which the true membrane potential $V(t)$ exceeds a reasonable threshold $V_{\text{th}}$. With a sufficiently small time step (we adopted $\delta t = 0.1$ [msec]), a single spike is transformed into a succession of spike occurrences $z_t = 1$. The mutual information computed according to Eq.(7) is depicted in Fig. 2B whose maximum position of $s \approx 2$ [msec] determines the optimal time shift.

The spike is predicted if the estimated probability $p_{\text{spike}}(t|\vec{v}_{t-s})$ of Eq.(10) exceeds a certain threshold value. Though it would be more efficient to use the systematic method suggested by Paninski et

al [16], we determined the threshold value empirically so that the coincidence factor $\Gamma$ described in the following is maximized.

Figure 3 compares a naive thresholding method and our state space method, in reference to the original spike times. It is observed from this figure that the prediction of the state space method is more accurate and robust than that of thresholding method.

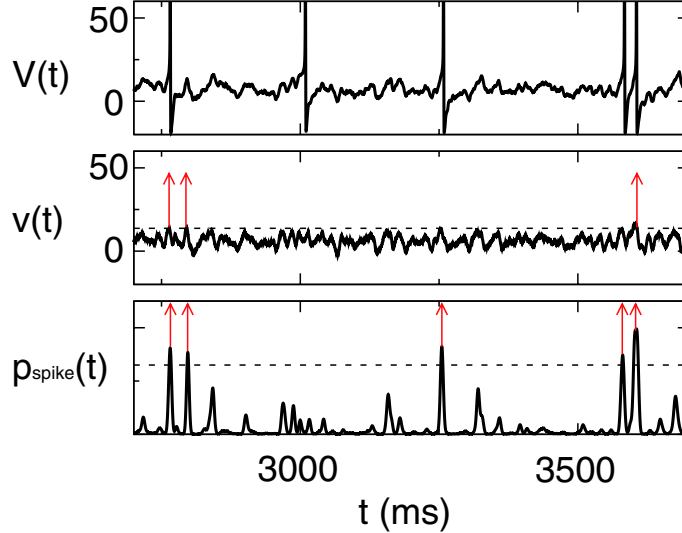

Figure 3: Comparison of the spike time predictions. (Top): The target membrane potential $V(t)$. (Middle): Prediction by thresholding the model potential. (Bottom): Prediction by the present state space method. Vertical arrows represent the predicted spikes.

Figure 4 depicts an orbit in the state space of $(V, V')$ of a target neuron for an instance of the spike generation, and the orbit of the predictive model in the state space of $(v, v')$ that mimics it. The predictive model can mimic the target orbit in the subthreshold region, but fails to catch the spiking orbit in the suprathreshold region. The spike occurrence is predicted by estimating the conditional probability, Eq.(10), given the state $(v, v')$ of the predictive model.

Contour lines of the probability in Figure 4C are not parallel to the $dv/dt$-axis. The contour lines for higher probabilities of spiking resemble an ad hoc "dynamic spike threshold" introduced by Azouz and Gray [17]. Namely, $v$ drops with $dv/dt$ along the contour lines. Contrastingly, the contour lines for lower probabilities of spiking are inversely curved: $v$ increases with $dv/dt$ along the contour lines. In the present framework, the state space information corresponding to the relatively low probability of spiking is effectively used for predicting spike times.

Prediction performance is compared with the benchmark of Kistler et al [8], the "coincidence factor,"

$$\Gamma(\Delta) = \frac{N_{\text{coinc}} - \langle N_{\text{coinc}} \rangle}{\frac{1}{2}(N_{\text{data}} + N_{\text{model}})} \frac{1}{1 - 2\nu\Delta}, \tag{12}$$

where $N_{\text{data}}$ and $N_{\text{model}}$ respectively represent the numbers of spikes in the original data and prediction model, $N_{\text{coinc}}$ is the number of coincident spikes with the precision of $\Delta$, $\langle N_{\text{coinc}} \rangle = 2\nu\Delta N_{\text{data}}$ is the expected number of coincidences of the data and the Poisson spikes with rate $\nu$. $\Delta$ is chosen as 2 [msec] in accordance with Jolivet et al [10].

Table 1: The coincidence factors evaluated for two methods of prediction based on the linear filter model.

| method | Currents I | Currents II |
|---|---|---|
| thresholding | 0.272 | 0.567 |
| state space | 0.430 | 0.666 |

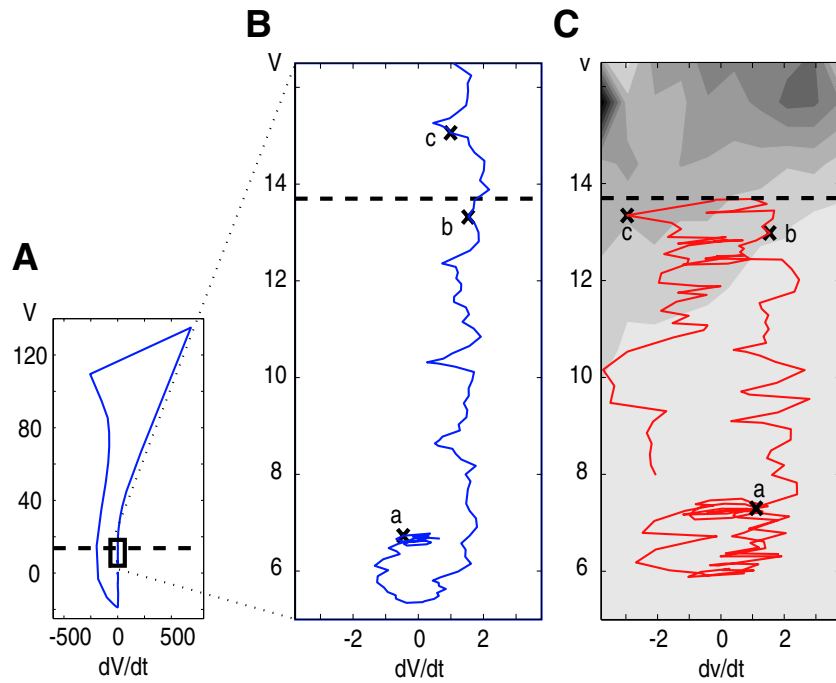

Figure 4: A: An orbit in the state space of $(V, V')$ of a target neuron for an instance of the spike generation (from 3240 to 3270 [msec] of Fig. 3). B: magnified view. C: The orbit in the state space of $(v, v')$ of the predictive model that mimics the target neuron. Contours represent the probability of spike occurrence computed with the Bayes formula, Eq.(10).The dashed lines represent the threshold adopted in the naive thresholding method (Fig 3 Middle). Three points $a$, $b$, and $c$ in the spaces of $(V, V')$ and $(v, v')$ represent the states of identical times, respectively, $t = 3242$, $3252$ and $3253$ [msec].

Table 2: The coincidence factors evaluated for two methods of prediction based on the spike response model.

| method | Currents I | Currents II |
|---|---|---|
| thresholding | 0.501 | 0.805 |
| state space | 0.641 | 0.842 |

The coincidence factors evaluated for a simple thresholding method and the state space method based on the linear filter model are summarized in Table 1, and those based on the spike response model are summarized in Table 2. It is observed that the prediction is significantly improved by our state space method.

It should be noted, however, that a model with the same set of parameters does not perform well over a range of inputs generated with different mean and variance: The model parameterized with the Currents I does not effectively predict the spikes of the neuron for the Current II, and vice versa. Nevertheless, our state-space method exhibits the better prediction than the naive thresholding strategy, if the statistics of the different inputs are relatively similar.

## 4 Summary

We proposed a method of evaluating the probability of the spike occurrence by observing the state space of the membrane potential and its time derivative(s) in advance of the possible spike time. It is found that the prediction is significantly improved by the state space method compared to the prediction obtained by simply thresholding an instantaneous value of the estimated potential. It is interesting to apply our method to biological data and categorize neurons based on their spiking mechanisms.

The state space method developed here is a rather general framework that may be applicable to any nonlinear phenomena composed of locally predictable dynamics. The generalization of linear filter analysis developed here has a certain similarity to the Linear-Nonlinear-Poisson (LNP) model [18, 19]. It would be interesting to generalize the present method of analysis to a wider range of phenomena such as the analysis of the coding of visual system [19, 20].

## Acknowledgments

This study is supported in part by Grants-in-Aid for Scientific Research to SS from the Ministry of Education, Culture, Sports, Science and Technology of Japan (16300068, 18020015) and the 21st Century COE "Center for Diversity and Universality in Physics" and to RK from Foundation For C & C Promotion.

## Appendix: Fast-spiking neuron model

The fast-spiking neuron model proposed by Erisir et al [13] was used in this contribution as a (virtual) target experiment. The details of the model were adjusted to Jolivet et al [10] to allow the direct comparison of the performances. Specifically, the model is described as

$$C\frac{du(t)}{dt} = -[I_{\text{Na}} + I_{\text{K}_1} + I_{\text{K}_2} + I_{\text{L}}] + I^{\text{ext}}(t) \,, \tag{13}$$

$$I_{\text{Na}} = g_{\text{Na}}m^3 h(u - E_{\text{Na}}) \,, \tag{14}$$

$$I_{\text{K}_1} = g_{\text{K}_1}n_1^4(u - E_{\text{K}}) \,, \qquad I_{\text{K}_2} = g_{\text{K}_2}n_2^2(u - E_{\text{K}}) \,, \tag{15}$$

$$I_{\text{L}} = g_{\text{L}}(u - E_{\text{L}}) \,, \tag{16}$$

where the gate variables $x \equiv n_1, n_2, m$ and $h$ obey the differential equations of the form,

$$\frac{dx}{dt} = \alpha_x(u)(1 - x) - \beta_x(u)x, \tag{17}$$

whose parameters $\alpha_x(u)$ and $\beta_x(u)$ are functions of $u$, as listed in Table 3.

Table 3: The parameters for the fast-spiking model. The membrane capacity is $C = 1.0[\mu\mathrm{F}/\mathrm{cm}^2]$.

| Channel | Variable | $\alpha$ | $\beta$ | $g_x(\mathrm{mS}/\mathrm{cm}^2)$ | $E_x(\mathrm{mV})$ |
|---------|----------|----------|---------|----------------------------------|--------------------|
| Na | m | $\dfrac{-3020+40u}{1-\exp(-\frac{u-75.5}{13.5})}$ | $\dfrac{1.2262}{\exp(\frac{u}{42.248})}$ | 112.5 | 74 |
| $--$ | h | $\dfrac{0.0035}{\exp(\frac{u}{24.186})}$ | $\dfrac{0.8712+0.017u}{1-\exp(-\frac{51.25+u}{5.2})}$ | $--$ | $--$ |
| $K_1$ | $n_1$ | $\dfrac{0.014(44+u)}{1-\exp(-\frac{44+u}{2.3})}$ | $\dfrac{0.0043}{\exp(\frac{44+u}{34})}$ | 0.225 | $-90.0$ |
| $K_2$ | $n_2$ | $\dfrac{u-95}{1-\exp(-\frac{u-95}{11.8})}$ | $\dfrac{0.025}{\exp(\frac{u}{22.22})}$ | 225.0 | $-90.0$ |
| $L$ | $--$ | $--$ | $--$ | 0.25 | $-70$ |

**References**

[1] Hodgkin, A.L. & Huxley, A.F. (1952) *J. Physiol.* **117**:500-544.

[2] FitzHugh, R. (1961) *Biophys. J.* **1**:445-466.

[3] Nagumo, J., Arimoto, S. & Yoshizawa, S. (1962) *Proc. IRE* **50**:2061-2070.

[4] Lapicque, L. (1907) *J. Physiol. Pathol. Gen.* **9**:620-635.

[5] Hines, M.L. & Carnevale, N.T. (1997) *Neural Comp.* **9**:1179-1209.

[6] Bower, J.M. & Beeman, D. (1995) *The Book of GENESIS: Exploring Realistic Neural Models with the GEneral NEural SImulation System.* New York: Springer-Verlag.

[7] Tsubo, Y., Kaneko, T. & Shinomoto, S. (2004) *Neural Networks* **17**:165-173.

[8] Kistler, W., Gerstner, W. & van Hemmen, J.L. (1997) *Neural Comp.* **9**:1015-1045.

[9] Gerstner, W. & Kistler, W. (2002) *Spiking Neuron Models: Single Neurons, Populations, Plasticity.* Cambridge: Cambridge Univ. Press.

[10] Jolivet, R., Lewis, T.J. & Gerstner, W. (2004) *J. Neurophysiol.* **92**:959-976.

[11] Jolivet, R., Rauch, A., Lüscher, H.R. & Gerstner, W. (2006) Integrate-and-Fire models with adaptation are good enough: predicting spike times under random current injection. In Y. Weiss, B. Schölkopf and J. Platt (eds.), *Advances in Neural Information Processing Systems 18*, pp. 595-602. Cambridge, MA: MIT Press.

[12] Westwick, D.T. & Kearney, R.E. (2003) *Identification of Nonlinear Physiological Systems. (Ieee Press Series in Biomedical Engineering)* Piscataway: Wiley-IEEE Press.

[13] Erisir, A., Lau, D., Rudy, B. & Leonard, C.S. (1999) *J. Neurophysiol.* **82**:2476-2489.

[14] Mainen, Z.F. & Sejnowski, T.J. (1995) *Science* **268**:1503-1506.

[15] MacKay, D. (2003) *Information Theory, Inference and Learning Algorithms.* Cambridge: Cambridge Univ. Press.

[16] Paninski, L., Pillow, J.W. & Simoncelli, E.P. (2005) *Neurocomputing* **65-66**: 379-385

[17] Azouz, R. & Gray, C.M. (2000) *PNAS* **97**(14):8110-8115.

[18] Chichilnisky, E.J. (2001) *Network* **12**(2):199-213.

[19] Pillow, J.W., Paninski, L., Uzzell, V.J., Simoncelli, E.P. & Chichilnisky, E.J. (2005) *Journal of Neuroscience* **25**(47):11003-11013.

[20] Arcas, B.A., Fairhall, A.L. & Bialek, W. (2003) *Neural Comp.* **15**:1715-1749.
